# A Neuromorphic VLSI System for Modeling the Neural Control of Axial Locomotion

**Girish N. Patel**
girish@ece.gatech.edu

**Edgar A. Brown**
ebrown@ece.gatech.edu

**Stephen P. DeWeerth**
steved@ece.gatech.edu

School of Electrical and Computer Engineering
Georgia Institute of Technology
Atlanta, Ga. 30332-0250

## Abstract

We have developed and tested an analog/digital VLSI system that models the coordination of biological segmental oscillators underlying axial locomotion in animals such as leeches and lampreys. In its current form the system consists of a chain of twelve pattern generating circuits that are capable of arbitrary contralateral inhibitory synaptic coupling. Each pattern generating circuit is implemented with two independent silicon Morris–Lecar neurons with a total of 32 programmable (floating-gate based) inhibitory synapses, and an asynchronous address-event interconnection element that provides synaptic connectivity and implements axonal delay. We describe and analyze the data from a set of experiments exploring the system behavior in terms of synaptic coupling.

## 1 Introduction

In recent years, neuroscientists and modelers have made great strides towards illuminating structure and computational properties in biological motor systems. For example, much progress has been made toward understanding the neural networks that elicit rhythmic motor behaviors, including leech heartbeat, crustacean stomatogastric mill and lamprey swimming (a good review on these is in [1] and [2]). It is thought that these same mechanisms form the basis for more complex motor behaviors. The neural substrate for these control mechanisms are called central pattern generators (CPG). In the case of locomotion these circuits are distributed along the body (in the spinal cord of vertebrates or in the ganglia of invertebrates) and are richly interactive with sensory input and descending connections from the brain, giving rise to a highly distributed system as shown in Figure 1. In cases in which axial locomotion is involved, such as leech and lamprey swimming, synaptic interconnection patterns among autonomous segmental oscillators along the animal's axis produce coordinated motor patterns. These *intersegmental coordination* architectures have been well studied through both physiological experimentation and mathematical modeling. In addition, undulatory gaits in snakes have also been studied from a robotics perspective [3]. However, a thorough understanding of the computational principles in these systems is still lacking.

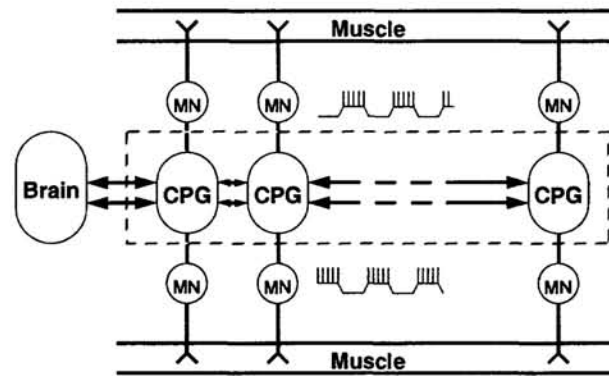

**Figure 1:** Neuroanatomy of segmented animals.

In order to better understand the computational paradigms that mediate intersegmental coordination and the resulting neural control of axial locomotion (and other motor patterns), we are using neuromorphic very large-scale integrated (VLSI) circuits to develop models of these biological systems. The goals in our research are (i) to study how the properties of individual neurons in a network affect the overall system behavior; (ii) to facilitate the validation of the principles underlying intersegmental coordination; and (iii) to develop a real-time, low power, motion control system. We want to exploit these principles and architectures both to improve our understanding of the biology and to design artificial systems that perform autonomously in various environments.

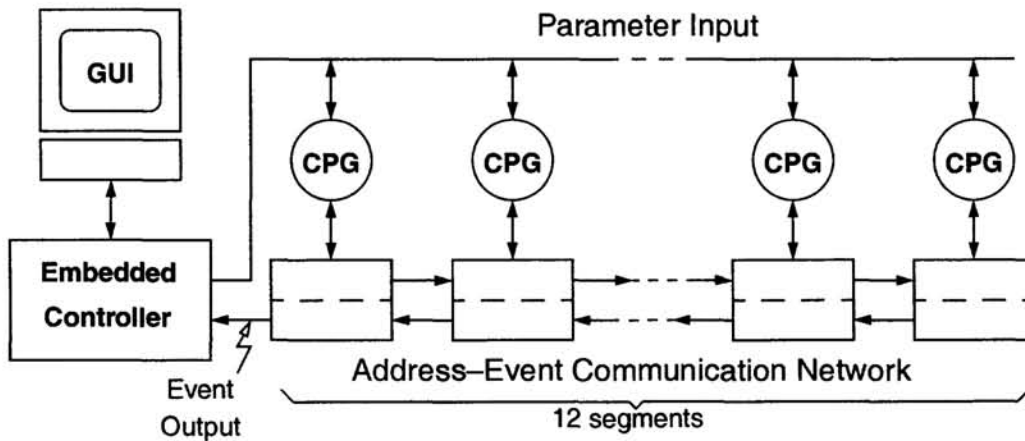

**Figure 2:** Block-level diagram of the implemented system. The intersegmental communications network facilitates communication among the intrasegmental units with pipelined stages.

In this paper, we present a VLSI model of intersegmental coordination as shown in Figure 2. Each segment in our system is implemented with a custom IC containing a CPG consisting of two silicon model neurons, each one with 16 inhibitory synapses whose values are stored on chip and are continuously variable; an asynchronous address event communications IC that implements the queuing and delaying of events providing synaptic connectivity and thus simulating axonal properties; and a microcontroller (with internal A/D converter and timer) that facilitates the modification of individual parameters through a serial bus. The entire system consists of twelve such segments linked to a computer on which a graphical user interface (GUI) is implemented. By using the GUI, we are able to control all of the synaptic connections in the system and to measure the result-

ing neural outputs. We present the system model, and we investigate the role of synaptic coupling in the establishment of phase lags along this chain of neural oscillators.

## 2 Pattern generating circuits

The smallest neural system capable of generating the basic alternating activity that characterizes the swimming CPGs is the *half-center* oscillator, essentially two bursting neurons with reciprocally inhibitory connections [1] as shown in Figure 3a. In biological systems, the associated neurons have both slow and fast time constants to facilitate the fast spiking (action potentials) and the slower bursting oscillations that control the elicited movements as shown in Figure 3b. To simplify the parameter space of our system, we use reduced two-state silicon neurons [4]. The output of each silicon neuron is an oscillation that represents the envelope of the bursting activity (i.e. the spiking activity and corresponding fast time constants are eliminated) as shown in Figure 3c. Each neuron also has 16 analog synapses that receive off-chip input. The synaptic parameters are stored in an array of floating-gate transistors [5] that provide nonvolatile analog memory.

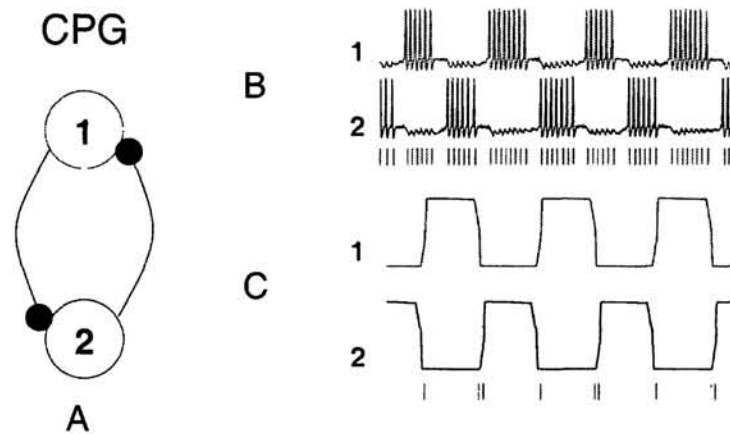

**Figure 3:** Half-center oscillator and the generation of events in spiking and nonspiking silicon neurons. Events are generated by detecting rapid rises in the membrane potential of spiking neurons or by detecting rapid rises and falls in nonspiking neurons.

## 3 Intersegmental communication

Our segmented system consists of an array of CPG circuits interconnected via an communication network that implements an asynchronous, address–event protocol [6][7]. Each CPG is connected to one node of this address–event intersegmental communication system as illustrated in Figure 2. This application-specific architecture uses a pipelined broadcast scheme that is based upon its biological counterpart. The principal advantage of using this custom scheme is that requisite addresses and delays are generated implicitly based upon the system architecture. In particular the system implements distance-dependent delays and relative addressing. The delays, which are thought to be integral to the network computation, replicate the axonal delays that result as action potentials propagate down an animal's body [2]. The relative addressing greatly simplifies the implementation of synaptic spread [8], the hypothesized translational invariance in the intersegmental connectivity in biological axial locomotion systems. Thus, we can set the synaptic parameters identically at every segment, greatly reducing system complexity.

In this architecture (which is described in more depth in [4]), each event is passed from segment to neighboring segment bidirectionally down the length of the one-dimensional

communications network. By delaying each event at every segment, the pipeline architecture facilitates the creation of distance-dependent delays. The other primary advantage of this architecture is that it can easily generate a relative addressing scheme. Figure 4 illustrates the event-passing architecture with respect to the relative addressing and distance-dependent delays. Each event, generated at a particular node (the center node, in this example), is transmitted bidirectionally down the length of the network. It is delayed by time $\Delta T$ at each segment, not including the initiating segment.

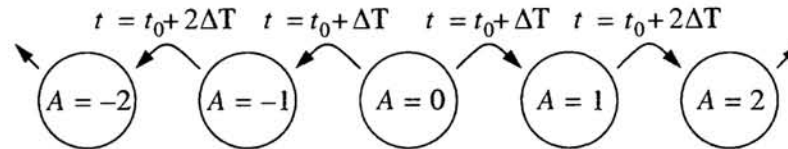

**Figure 4:** Relative addressing and distance-dependent delays.

The events are generated by the neurons in each segment. Because these are not spiking neurons, we could not use the typical scheme of generating one event per action potential. Instead, we generate one event at the beginning and end of each burst (as illustrated in Figure 3) and designate the individual events as rising or falling. In each segment the events are stored in a queue (Figure 5), which implements delay based upon uniform conduction velocities. As an event arrives at each new segment, it is time stamped, its relative address is incremented (or decremented), and then it is stored in the queue for the $\Delta T$ interval. As the event exits the queue, its data is decoded by the intrasegmental units, and synaptic inputs are applied to the appropriate intrasegmental neurons.

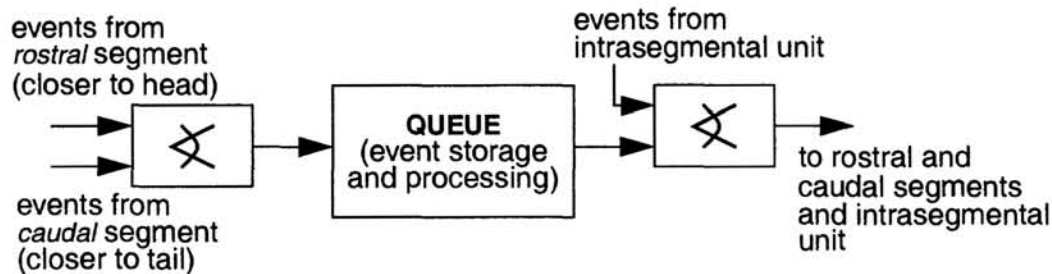

**Figure 5:** Block-level diagram of a communications node illustrating how events enter and exit each stage of the pipeline.

## 4  Experiments and Discussion

We have implemented the complete system shown in Figure 2, and have performed a number of experiments on the system. In Figure 6, we show the behaviors the system exhibits when it is configured with asymmetrical nearest-neighbor connections. The system displays traveling waves whose directions depend on the direction of the dominant coupling. Note that the intersegmental phase lags vary for different swim frequencies.

One important set of experiments focussed on the role of long-distance connections on the system behaviors. In these experiments, we configured the system with strong descending (towards the tail) connections such that robust rearward traveling waves (forward swimming) are observed. The long-distance connections are weak enough to avoid any *bifurcations* in behavior (different type of behavior). Thus, the traveling wave solution resulting from the nearest-neighbor connections persists as we progressively add long-distance connections. In Figure 7 we show the dependency of the swim frequency and the total phase lag (summation of the normalized intersegmental phase lags, where $1 \equiv 360°$) on the extent of the connections. The results show a clear difference in behav-

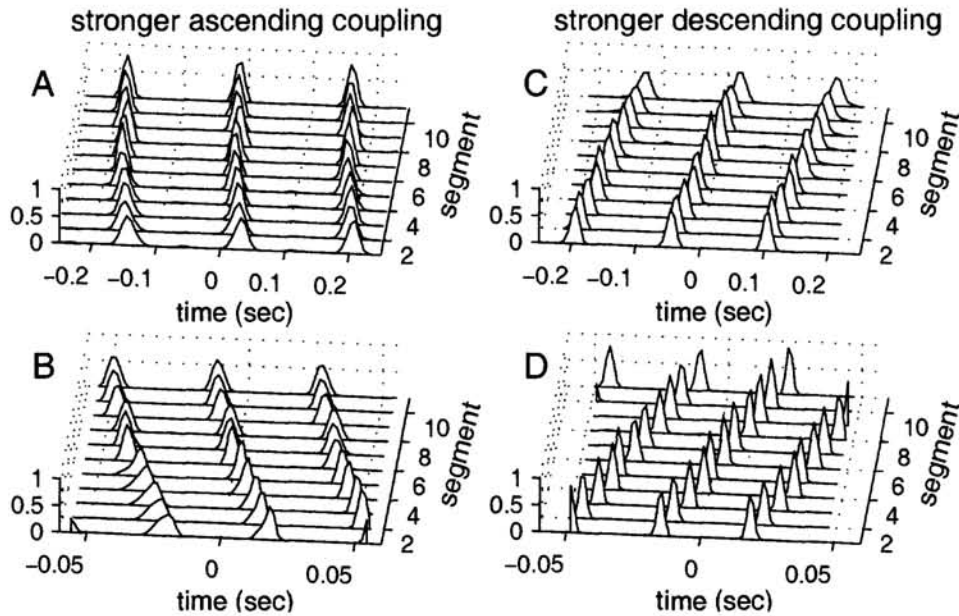

**Figure 6:** Traveling waves in the system with asymmetrical, nearest-neighbor connections. Plots are cross-correlations between rising edge events generated by a neuron in segment six and events generated by homolog neurons in each segment. Stronger ascending connections (A & B) produce forward traveling waves (backward swimming) and stronger descending connections (C & D) produce rearward traveling waves (forward swimming). An externally applied current ($I_{ext}$) controls the swim frequency. At small values of $I_{ext}$ (6.7 nA) the periods of the swim cycles are approximately 0.180 ms and 0.150 ms for A & C, respectively; for large values of $I_{ext}$ (32.8 nA), the periods of the swim cycles are approximately 36 ms and 33 ms for B & D, respectively.

iors between the lowest tonic drive ($I_{ext}$ = 21.9 nA) and the two higher tonic drives. (By tonic drive, we mean a constant dc current is applied to all neurons.) In the former, the sensitivity of long-distance connections on frequency and intersegmental phase lags is considerably greater than in the latter. The demarcation in behavior may be attributed to different behaviors at different tonic drives. For lower tonic drive, the long-distance connections tend to synchronize the system (decrease the intersegmental phase lags). At the higher tonic drives, long-distance connections do not affect the system considerably. For $I_{ext}$ = 32.8 nA, connections that span up to four segments aid in producing uniformity in the intersegmental phase lags. Although this does not hold for $I_{ext}$ = 48.1 nA, long-distance connections play a more significant role in preserving the total phase difference. At $I_{ext}$ = 32.8 nA and $I_{ext}$ = 48.1 nA, the system with short-distance connections produces a total phase difference of 1.19 and 1.33, respectively. In contrast, for $I_{ext}$ = 32.8 nA and $I_{ext}$ = 48.1 nA, the system with long-distance connections that span up to seven segments produces a total phase difference of 1.20 and 1.25, respectively.

In the above experiments, we have demonstrated that, in a specific parameter regime, weak long-distance connections can affect the intersegmental phase lags. However, these weight profiles should not be construed as a possible explanation on what the weight profiles in a biological system might be. The parameter regime in which we observed this behavior is small; at moderate strengths of coupling, the traveling wave solutions disappear and move towards synchronous behavior. Recent experiments done on spinalized lampreys reveal that long-distance connections are moderately strong [10]. Thus, our current model is unable to replicate this aspect of intersegmental coordination. There are several explanations that may account for this discrepancy.

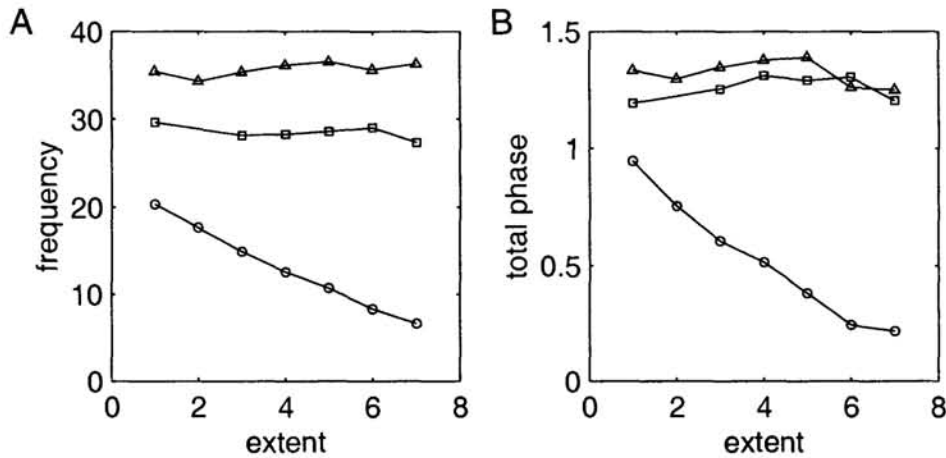

Figure 7: Effects of weak long-distance connections on swimming frequency (A), on the total phase difference (summation of the normalized intersegmental phase lags) (B), and on the standard deviation of the intersegmental phase lags (C). $5 < =$ denote $I_{ext} = 48.1$ nA, 32.8 nA, and 21 nA, respectively.

In the segmental CPG network of the animal, there are many classes of neurons that send projections to many other classes of neurons. The phase a connection imposes is determined by which neuron class connects with which other neuron class. In our system, the segmental CPG network has only a single class of neurons upon which the long-distance connections can impose their phase. Depending on where in parameter space we operate our system, the long-distance connections have too little or too great an effect on the behavior of the system. At high tonic drives, the sensitivity of the weak long-distance connections on the intersegmental phase lags is small, whereas for small tonic drives, the long-distance connections have a great effect on the intersegmental phase lags.

It has been shown that if the waveform of the oscillators is sinusoidal (i.e., the time scales of the two state variables are not too different), traveling wave solutions exist and have a large basin of attraction [11]. However, as the disparity between the two time scales is made larger (i.e., the neurons are stiff and the waveform of the oscillations appears square-wave like), the system will move towards synchrony. In our implementation, to facilitate accurate communication of events, we bias the neurons with relatively large differences in the time scales. Thus, this restriction reduces the parameter regime in which we can observe stable traveling waves.

Another factor that determines the range of parameters in which stable traveling waves are observed is the slope of our synaptic coupling function. When the slope of the coupling function is steep, the total synaptic current over a cycle can increase significantly, causing weak connections to appear strong. This has an overall effect of synchronizing the network [11]. For coupling functions whose slopes are shallow, the total synaptic current over a cycle is reduced; therefore, the connections appear weak and larger intersegmental phase lags are possible. Thus, the sharp synaptic coupling function in our implementation, which is necessary for communication, is another factor that diminishes the parameter regime in which we can observe stable traveling waves.

The above factors limit the parameter range in which we observe traveling waves. However, all of these issues can be addressed by improving our CPG network. The first issue can be addressed by increasing the number of neuron classes or adding more segments. The second and third issues can be addressed by adding spiking neurons in our CPG network so that the form of the oscillations can be coded in the spike train and the synaptic coupling functions can be implemented on the receiving side of the CPG chip. The fourth

issue can be addressed by designing self-adapting neurons that tune their internal parameters so that their waveforms and intrinsic frequencies are matched. Although weak coupling may not be biologically plausible, producing traveling waves based on phase oscillators would be an interesting research direction.

## 5 Conclusions and Future Work

In this paper, we described a functional, neuromorphic VLSI system that implements an array of neural oscillators interconnected by an address–event communication network. This system represents our most ambitious neuromorphic VLSI effort to date, combining 24 custom ICs, a special-purpose asynchronous communication architecture designed analogously to its biological counterpart, large-scale synaptic interconnectivity with parameters stored using floating-gate devices, and a computer interface for setting the parameters and for measuring the neural activity. The working system represents the culmination of a four-year effort, and now provides a testbed for exploring a variety of biological hypotheses and theoretical predictions.

Our future directions in the development of this system are threefold. First, we will continue to explore, in depth, the operation of the present system, comparing it to theoretical predictions and biological hypotheses. Second, we are implementing a segmented mechanical system that will provide a moving output and will facilitate the implementation of sensory feedback. Third, we are developing new CPG model centered around sensory feedback and motor learning. The modular design of the system, which puts all of the neural and synaptic specificity on the CPG IC, allows us to design a completely new CPG and to replace it in the system without changing the communication architecture.

## References

[1] E. Marder & R.L. Calabrese. Principles of rhythmic motor pattern generation. *Physiological Reviews* 76 (3): 687–717, 1996.

[2] A.H. Cohen, G.B. Ermentrout, T. Kiemel, N. Kopell, K.A. Sigvardt, & T.L. Williams. Modeling of intersegmental coordination in the lamprey central pattern generator for locomotion. *TINS* 15:434–438, 1992.

[3] S. Hirose. *Biologically Inspired Robots: Snake-like Locomotors and Manipulators.* Oxford University Press, 1993.

[4] S. DeWeerth, G. Patel, D. Schimmel, M. Simoni, & R.L. Calabrese. A VLSI Architecture for Modeling Intersegmental Coordination. In *Proceedings of the Seventeenth Conference on Advanced Research in VLSI*, R.B. Brown and A.T. Ishii (eds), Los Alamitos, CA: IEEE Computer Society, 182–200, 1997.

[5] P. Hasler, B.A. Minch, and C. Diorio. Adaptive circuits using *p*Fet floating-gate devices. In Scott Wills and Stephen DeWeerth editors, *20th Conference of Advanced Research in VLSI*, pages 215–230, Los Alamitos, California, CA: IEEE Computer Society, 1999.

[6] M.A. Mahowald. VLSI Analogs of Neuronal Visual Processing: A Synthesis of Form and Function. *Ph.D. Thesis, California Institute of Technology*, Pasadena, CA, 1992.

[7] K.A. Boahen. Communicating Neuronal Ensembles between Neuromorphic Chips. *Analog Integrated Circuits and Signal Processing*, 1997.

[8] T. Willams. Phase Coupling and Synaptic Spread in Chains of Coupled Neuronal Oscillators. *Science*, vol. 258, 1992.

[9] G. Patel. A Neuromorphic Architecture for Modeling Intersegmental Coordination. *Ph.D. Thesis, Georgia Institute of Technology*, Atlanta, GA, 1999.

[10] A. H. Cohen. Personal communication.

[11] D. Somers & N. Kopell. Waves and synchrony in networks of oscillators of relaxation and non-relaxation type. *Phyica D*, 89:169–183, 1995.

[12] N. Kopell & G.B. Ermentrout. Coupled oscillators and the design of central pattern generators. *Mathematical Biosciences*, 90:87–109, 1988.